# Robotic Grasping of Novel Objects

**Ashutosh Saxena,   Justin Driemeyer,   Justin Kearns,   Andrew Y. Ng**
Computer Science Department
Stanford University, Stanford, CA 94305
{asaxena,jdriemeyer,jkearns,ang}@cs.stanford.edu

## Abstract

We consider the problem of grasping novel objects, specifically ones that are being seen for the first time through vision. We present a learning algorithm that neither requires, nor tries to build, a 3-d model of the object. Instead it predicts, directly as a function of the images, a point at which to grasp the object. Our algorithm is trained via supervised learning, using synthetic images for the training set. We demonstrate on a robotic manipulation platform that this approach successfully grasps a wide variety of objects, such as wine glasses, duct tape, markers, a translucent box, jugs, knife-cutters, cellphones, keys, screwdrivers, staplers, toothbrushes, a thick coil of wire, a strangely shaped power horn, and others, none of which were seen in the training set.

## 1   Introduction

In this paper, we address the problem of grasping novel objects that a robot is perceiving for the first time through vision.

Modern-day robots can be carefully hand-programmed or "scripted" to carry out many complex manipulation tasks, ranging from using tools to assemble complex machinery, to balancing a spinning top on the edge of a sword. [15] However, autonomously grasping a previously unknown object still remains a challenging problem. If the object was previously known, or if we are able to obtain a full 3-d model of it, then various approaches, for example ones based on friction cones, [5] form- and force-closure, [1] pre-stored primitives, [7] or other methods can be applied. However, in practical scenarios it is often very difficult to obtain a full and accurate 3-d reconstruction of an object seen for the first time through vision. This is particularly true if we have only a single camera; for stereo systems, 3-d reconstruction is difficult for objects without texture, and even when stereopsis works well, it would typically reconstruct only the visible portions of the object. Finally, even if more specialized sensors such as laser scanners (or active stereo) are used to estimate the object's shape, we would still have only a 3-d reconstruction of the front face of the object.

In contrast to these approaches, we propose a learning algorithm that neither requires, nor tries to build, a 3-d model of the object. Instead it predicts, directly as a function of the images, a point at which to grasp the object. Informally, the algorithm takes two or more pictures of the object, and then tries to identify a point within each 2-d image that corresponds to a good point at which to grasp the object. (For example, if trying to grasp a coffee mug, it might try to identify the mid-point of the handle.) Given these 2-d points in each image, we use triangulation to obtain a 3-d position at which to actually attempt the grasp. Thus, rather than trying to triangulate every single point within each image in order to estimate depths—as in dense stereo—we only attempt to triangulate one (or at most a small number of) points corresponding to the 3-d point where we will grasp the object. This allows us to grasp an object without ever needing to obtain its full 3-d shape, and applies even to textureless, translucent or reflective objects on which standard stereo 3-d reconstruction fares poorly.

To the best of our knowledge, our work represents the first algorithm capable of grasping novel objects (ones where a 3-d model is not available), including ones from novel object classes, that we are perceiving for the first time using vision.

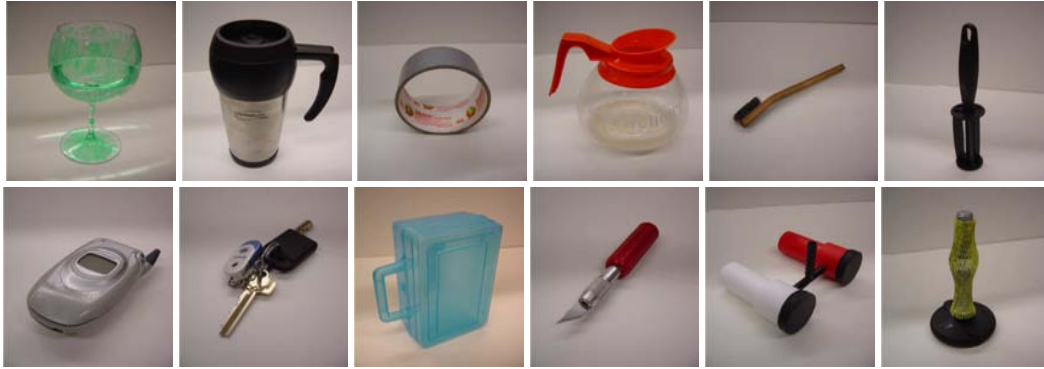
Figure 1: Examples of objects on which the grasping algorithm was tested.

In prior work, a few others have also applied learning to robotic grasping. [1] For example, Jebara et al. [8] used a supervised learning algorithm to learn grasps, for settings where a full 3-d model of the object is known. Hsiao and Lozano-Perez [4] also apply learning to grasping, but again assuming a fully known 3-d model of the object. Piater's algorithm [9] learned to position single fingers given a top-down view of an object, but considered only very simple objects (specifically, square, triangle and round "blocks"). Platt et al. [10] learned to sequence together manipulation gaits, but again assumed a specific, known, object. There is also extensive literature on recognition of known object classes (such as cups, mugs, etc.) [14], but this seems unlikely to apply directly to grasping objects from novel object classes.

To pick up an object, we need to identify the grasping point—more formally, a position for the robot's end-effector. This paper focuses on the task of grasp identification, and thus we will consider only objects that can be picked up without performing complex manipulation.[1] We will attempt to grasp a number of common office and household objects such as toothbrushes, pens, books, mugs, martini glasses, jugs, keys, duct tape, and markers. (See Fig. 1.)

The remainder of this paper is structured as follows. In Section 2, we describe our learning approach, as well as our probabilistic model for inferring the grasping point. In Section 3, we describe the motion planning/trajectory planning (on our 5 degree of freedom arm) for moving the manipulator to the grasping point. In Section 4, we report the results of extensive experiments performed to evaluate our algorithm, and Section 5 concludes.

## 2    Learning the Grasping Point

Because even very different objects can have similar sub-parts, there are certain visual features that indicate good grasps, and that remain consistent across many different objects. For example, jugs, cups, and coffee mugs all have handles; and pens, white-board markers, toothbrushes, screw-drivers, etc. are all long objects that can be grasped roughly at their mid-point. We propose a learning approach that uses visual features to predict good grasping points across a large range of objects.

Given two (or more) images of an object taken from different camera positions, we will predict the 3-d position of a grasping point. An image is a projection of the three-dimensional world into an image plane, and does not have depth information. In our approach, we will predict the 2-d location of the grasp in each image; more formally, we will try to identify the projection of a good grasping point onto the image plane. If each of these points can be perfectly identified in each image, we can then easily "triangulate" from these images to obtain the 3-d grasping point. (See Fig. 4a.) In practice it is difficult to identify the projection of a grasping point into the image plane (and, if there are multiple grasping points, then the correspondence problem—i.e., deciding which grasping point in one image corresponds to which point in another image—must also be solved). On our robotic platform, this problem is further exacerbated by uncertainty in the position of the camera when the

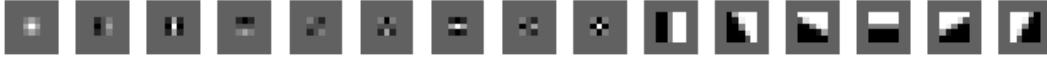

Figure 2: Examples of different edge and texture filters used to calculate the features.

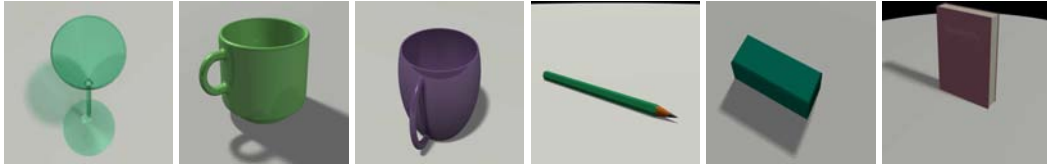

Figure 3: Synthetic images of the objects used for training. The classes of objects used for training were martini glasses, mugs, teacups, pencils, whiteboard erasers, and books.

images were taken. To address all of these issues, we develop a probabilistic model over possible grasping points, and apply it to infer a good position at which to grasp an object.[2]

## 2.1 Features

In our approach, we begin by dividing the image into small rectangular patches, and for each patch predict if it is a projection of a grasping point onto the image plane. For this prediction problem, we chose features that represent three types of local cues: edges, textures, and color. [11, 13] We compute features representing edges by convolving the intensity channel[3] with 6 oriented edge filters (Fig. 2). Texture information is mostly contained within the image intensity channel, so we apply 9 Laws masks to this channel to compute the texture energy. For the color channels, low frequency information is most useful to identify grasps; our color features are computed by applying a local averaging filter (the first Laws mask) to the 2 color channels. We then compute the sum-squared energy of each of these filter outputs. This gives us an initial feature vector of dimension 17.

To predict if a patch contains a grasping point, local image features centered on the patch are insufficient, and one has to use more global properties of the object. We attempt to capture this information by using image features extracted at multiple spatial scales (3 in our experiments) for the patch. Objects exhibit different behaviors across different scales, and using multi-scale features allows us to capture these variations. In detail, we compute the 17 features described above from that patch as well as the 24 neighboring patches (in a 5x5 window centered around the patch of interest). This gives us a feature vector $x$ of dimension $1*17*3 + 24*17 = 459$.

## 2.2 Synthetic Data for Training

We apply supervised learning to learn to identify patches that contain grasping points. To do so, we require a labeled training set, i.e., a set of images of objects labeled with the 2-d location of the grasping point in each image. Collecting real-world data of this sort is cumbersome, and manual labeling is prone to errors. Thus, we instead chose to generate, and learn from, synthetic data that is automatically labeled with the correct grasps.

In detail, we generate synthetic images along with correct grasp (Fig. 3) using a computer graphics ray tracer,[4] as this produces more realistic images than other simpler rendering methods.[5] The advantages of using synthetic images are multi-fold. First, once a synthetic model of an object has been created, a large number of training examples can be automatically generated by rendering the object under different (randomly chosen) lighting conditions, camera positions and orientations, etc.

In addition, to increase the diversity of the training data generated, we randomized different properties of the objects such as color, scale, and text (e.g., on the face of a book). The time-consuming part of synthetic data generation is the creation of the mesh models of the objects. However, there are many objects for which models are available on the internet, and can be used with only minor modifications. We generated 2500 examples from synthetic data, comprising objects from six object classes (see Figure 3). Using synthetic data also allows us to generate perfect labels for the training set with the exact location of a good grasp for each object. In contrast, collecting and manually labeling a comparably sized set of real images would have been extremely time-consuming.

## 2.3  Probabilistic Model

On our manipulation platform, we have a camera mounted on the robotic arm. (See Fig. 6) When asked to grasp an object, we command the arm to move the camera to two or more positions, so as to acquire images of the object from these positions. However, there are inaccuracies in the physical positioning of the arm, and hence some slight uncertainty in the position of the camera when the images are acquired. We now describe how we model these position errors. Formally, let $C$ be the image that would have been taken if the robot had moved exactly to the commanded position and orientation. However, due to robot positioning error, instead an image $\hat{C}$ is taken from a slightly different location. Let $(u, v)$ be a 2-d position in image $C$, and let $(\hat{u}, \hat{v})$ be the corresponding image position in $\hat{C}$. Thus $C(u, v) = \hat{C}(\hat{u}, \hat{v})$, where $C(u, v)$ is the pixel value at $(u, v)$ in image $C$. The errors in camera position/pose should usually be small,[6] and we model the difference between $(u, v)$ and $(\hat{u}, \hat{v})$ using an additive Gaussian model: $\hat{u} = u + \epsilon_u$, $\hat{v} = v + \epsilon_v$, where $\epsilon_u, \epsilon_v \sim N(0, \sigma^2)$.

For each location $(u, v)$ in an image $C$, we define the class label to be $z(u, v) = 1\{(u, v)$ is the projection of a grasping point into image plane$\}$. (Here, $1\{\cdot\}$ is the indicator function; $1\{\text{True}\} = 1$, $1\{\text{False}\} = 0$.) For a corresponding location $(\hat{u}, \hat{v})$ in image $\hat{C}$, we similarly define $\hat{z}(\hat{u}, \hat{v})$ to indicate whether position $(\hat{u}, \hat{v})$ represents a grasping point in the image $\hat{C}$. Since, $(u, v)$ and $(\hat{u}, \hat{v})$ are corresponding pixels in $C$ and $\hat{C}$, we assume $\hat{z}(\hat{u}, \hat{v}) = z(u, v)$. Thus:

$$P(z(u, v) = 1 | C) = P(\hat{z}(\hat{u}, \hat{v}) = 1 | \hat{C}) \tag{1}$$

$$= \int_{\epsilon_u} \int_{\epsilon_v} P(\epsilon_u, \epsilon_v) P(\hat{z}(u + \epsilon_u, v + \epsilon_v) = 1 | \hat{C}) d\epsilon_u d\epsilon_v \tag{2}$$

Here, $P(\epsilon_u, \epsilon_v)$ is the (Gaussian) density over $\epsilon_u$ and $\epsilon_v$. Further, we use logistic regression to model the probability of a 2-d position $(u + \epsilon_u, v + \epsilon_v)$ in $\hat{C}$ being a good grasping point:

$$P(\hat{z}(u + \epsilon_u, v + \epsilon_v) = 1 | \hat{C}) = P(\hat{z}(u + \epsilon_u, v + \epsilon_v) = 1 | x; w) = 1/(1 + e^{-w^T x}) \tag{3}$$

where $x \in \mathbb{R}^{459}$ are the features for the rectangular patch centered at $(u + \epsilon_u, v + \epsilon_v)$ in image $\hat{C}$ (described in Section 2.1). The parameter of this model $w \in \mathbb{R}^{459}$ is learned using standard maximum likelihood for logistic regression: $w = \arg\max_{w'} \prod_i P(z_i | x_i; w')$, where $(x_i, z_i)$ are the synthetic training examples (image patches and labels), as described in Section 2.2. Fig. 5 shows the result of applying the learned logistic regression model to some real (non-synthetic) images.

Given two or more images of a new object from different camera positions, we want to infer the 3-d position of the grasping point. (See Fig. 4.) Because logistic regression may have predicted multiple grasping points per image, there is usually ambiguity in the correspondence problem (i.e., which grasping point in one image corresponds to which graping point in another). To address this while also taking into account the uncertainty in camera position, we propose a probabilistic model over possible grasping points in 3-d space. In detail, we discretize the 3-d work-space of the robotic arm into a regular 3-d grid $G \subset \mathbb{R}^3$, and associate with each grid element $j$ a random variable $y_j = 1\{\text{grid cell } j \text{ is a grasping point}\}$.

From each camera location $i = 1, ..., N$, one image is taken. In image $C_i$, let the ray passing through $(u, v)$ be denoted $R_i(u, v)$. Let $G_i(u, v) \subset G$ be the set of grid-cells through which the ray $R_i(u, v)$ passes. Let $r_1, ... r_K \in G_i(u, v)$ be the indices of the grid-cells lying on the ray $R_i(u, v)$.

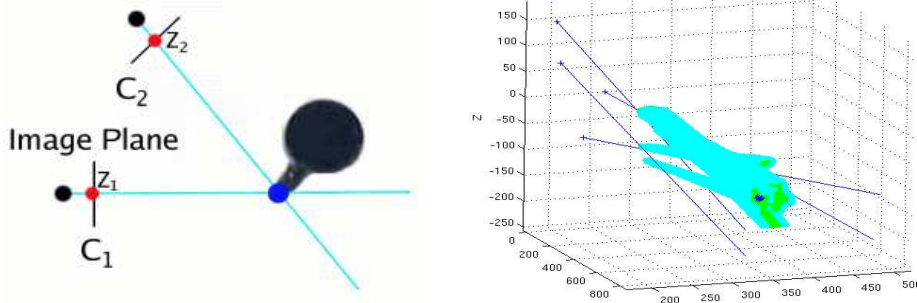

Figure 4: (a) Diagram illustrating rays from two images $C_1$ and $C_2$ intersecting at a grasping point (shown in dark blue). (b) Actual plot in 3-d showing multiple rays from 4 images intersecting at the grasping point. All grid-cells with at least one ray passing nearby are colored using a light blue-green-dark blue colormap, where dark blue represents those grid-cells which have many rays passing near them. (Best viewed in color.)

We know that if any of the grid-cells $r_j$ along the ray represent a grasping point, then its projection is a grasp point. More formally, $z_i(u,v) = 1$ if and only if $y_{r_1} = 1$ or $y_{r_2} = 1$ or $\ldots$ or $y_{r_K} = 1$. For simplicity, we use a (arguably unrealistic) naive Bayes-like assumption of independence, and model the relation between $P(z_i(u,v) = 1|C_i)$ and $P(y_{r_1} = 1$ or $\ldots$ or $y_{r_K} = 1|C_i)$ as

$$P(z_i(u,v) = 0|C_i) = P(y_{r_1} = 0, ..., y_{r_K} = 0|C_i) = \prod_{j=1}^{K} P(y_{r_j} = 0|C_i) \tag{4}$$

Assuming that any grid-cell along a ray is equally likely to be a grasping point, this therefore gives

$$P(y_{r_j} = 1|C_i) = 1 - (1 - P(z_i(u,v) = 1|C_i))^{1/K} \tag{5}$$

Next, using another naive Bayes-like independence assumption, we estimate the probability of a particular grid-cell $y_j \in G$ being a grasping point as:

$$P(y_j = 1|C_1, ..., C_N) = \frac{P(y_j=1)P(C_1,...,C_N|y_j=1)}{P(C_1,...,C_N)} = \frac{P(y_j=1)}{P(C_1,...,C_N)} \prod_{i=1}^{N} P(C_i|y_j = 1) \tag{6}$$

$$= \frac{P(y_j=1)}{P(C_1,...,C_N)} \prod_{i=1}^{N} \frac{P(y_j=1|C_i)P(C_i)}{P(y_j=1)} \propto \prod_{i=1}^{N} P(y_j = 1|C_i) \tag{7}$$

where $P(y_j = 1)$ is the prior probability of a grid-cell being a grasping point (set to a constant value in our experiments). Using Equations 2, 3, 5, and 7, we can now compute (up to a constant of proportionality that does not depend on the grid-cell) the probability of any grid-cell $y_j$ being a valid grasping point, given the images.

## 2.4 MAP Inference

We infer the best grasping point by choosing the 3-d position (grid-cell) that is most likely to be a valid grasping point. More formally, using Eq. 5 and 7, we will choose:

$$\arg\max_j \ \log P(y_j = 1|C_1, ..., C_N) = \arg\max_j \ \log \prod_{i=1}^{N} P(y_j = 1|C_i) \tag{8}$$

$$= \arg\max_j \ \sum_{i=1}^{N} \log \left(1 - (1 - P(z_i(u,v) = 1|C_i))^{1/K}\right) \tag{9}$$

where $P(z_i(u,v) = 1|C_i)$ is given by Eq. 2 and 3. A straightforward implementation that explicitly computes the sum above for every single grid-cell would give good grasping performance, but be extremely inefficient (over 110 seconds). For real-time manipulation, we therefore used a more efficient search algorithm in which we explicitly consider only grid-cells $y_j$ that at least one ray $R_i(u,v)$ intersects. Further, the counting operation in Eq. 9 is implemented using an efficient counting algorithm that accumulates the sums over all grid-cells by iterating over all the images $N$ and rays $R_i(u,v)$.[7] This results in an algorithm that identifies a grasping position in 1.2 sec.

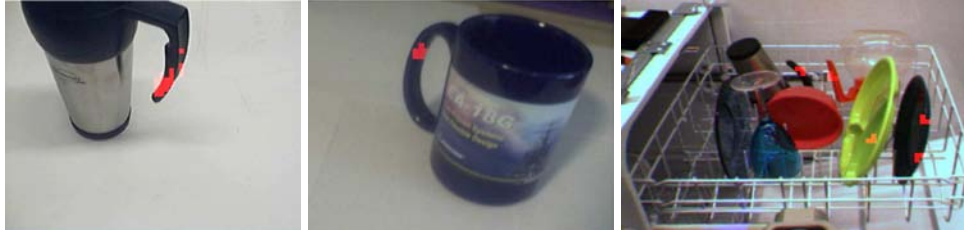

Figure 5: Grasping point classification. The red points in each image show the most likely locations, predicted to be candidate grasping points by our logistic regression model. (Best viewed in color.)

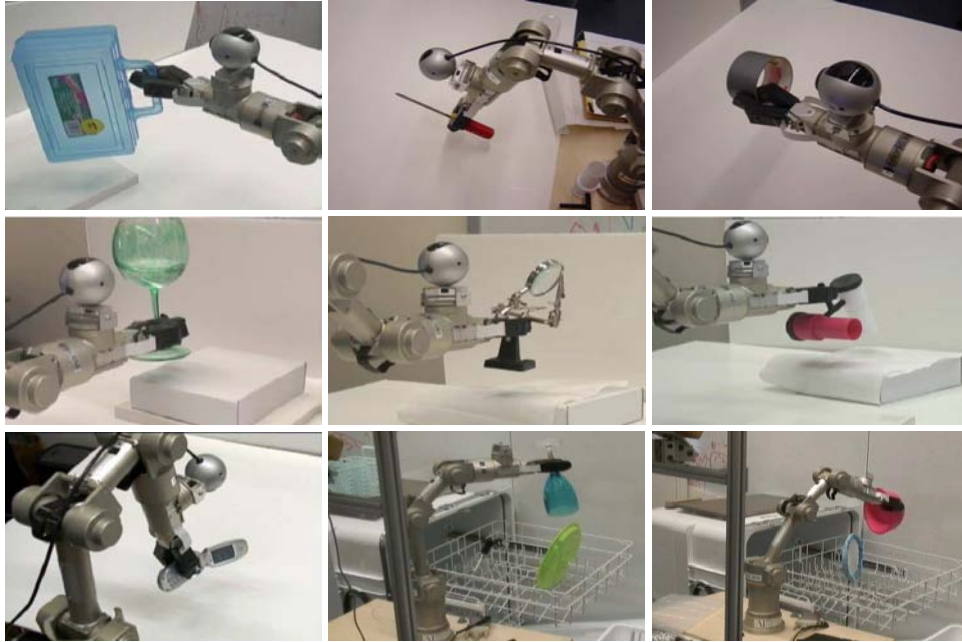

Figure 6: The robotic arm picking up various objects: box, screwdriver, duct-tape, wine glass, a solder tool holder, powerhorn, cellphone, and martini glass and cereal bowl from dishwasher.

## 3 Control

Having identified a grasping point, we have to move the end-effector of the robotic arm to it, and pick up the object. In detail, our algorithm plans a trajectory in joint angle space [5] to take the end-effector to an approach position,[8] and then moves the end-effecter in a straight line forward towards the grasping point. Our robotic arm uses two classes of grasps: *downward* grasps and *outward* grasps. These arise as a direct consequence of the shape of the workspace of our 5 dof robotic arm (Fig. 6). A "downward" grasp is used for objects that are close to the base of the arm, which the arm will grasp by reaching in a downward direction. An "outward" grasp is for objects further away from the base, for which the arm is unable to reach in a downward direction. The class is determined based on the position of the object and grasping point.

## 4 Experiments

### 4.1 Hardware Setup

Our experiments used a mobile robotic platform called STAIR (STanford AI Robot) on which are mounted a robotic arm, as well as other equipment such as our web-camera, microphones, etc. STAIR was built as part of a project whose long-term goal is to create a robot that can navigate home and office environments, pick up and interact with objects and tools, and intelligently converse with and help people in these environments. Our algorithms for grasping novel objects represent a first step towards achieving some of these goals. The robotic arm we used is the Harmonic Arm made by Neuronics. This is a 4 kg, 5-dof arm equipped with a parallel plate gripper, and has a positioning accuracy of $\pm 1$ mm. Our vision system used a low-quality webcam mounted near the end-effector.

Table 1: Average absolute error in locating the grasp point for different objects, as well as grasp success rate for picking up the different objects using our robotic arm. (Although training was done on synthetic images, testing was done on the real robotic arm and real objects.)

<table>
<tr><td colspan="3">OBJECTS SIMILAR TO ONES TRAINED ON</td><td colspan="3">NOVEL OBJECTS</td></tr>
<tr><td>TESTED ON</td><td>MEAN ERROR (CM)</td><td>GRASP-RATE</td><td>TESTED ON</td><td>MEAN ERROR (CM)</td><td>GRASP-RATE</td></tr>
<tr><td></td><td></td><td></td><td>DUCT TAPE</td><td>1.8</td><td>100%</td></tr>
<tr><td>MUGS</td><td>2.4</td><td>75%</td><td>KEYS</td><td>1.0</td><td>100%</td></tr>
<tr><td>PENS</td><td>0.9</td><td>100%</td><td>MARKERS/SCREWDRIVER</td><td>1.1</td><td>100%</td></tr>
<tr><td>WINE GLASS</td><td>1.2</td><td>100%</td><td>TOOTHBRUSH/CUTTER</td><td>1.1</td><td>100%</td></tr>
<tr><td>BOOKS</td><td>2.9</td><td>75%</td><td>JUG</td><td>1.7</td><td>75%</td></tr>
<tr><td>ERASER/</td><td></td><td></td><td>TRANSLUCENT BOX</td><td>3.1</td><td>75%</td></tr>
<tr><td>CELLPHONE</td><td>1.6</td><td>100%</td><td>POWERHORN</td><td>3.6</td><td>50%</td></tr>
<tr><td></td><td></td><td></td><td>COILED WIRE</td><td>1.4</td><td>100%</td></tr>
<tr><td>OVERALL</td><td>1.80</td><td>90%</td><td>OVERALL</td><td>1.85</td><td>87.5%</td></tr>
</table>

## 4.2 Results and Discussion

We first evaluated the predictive accuracy of the algorithm on synthetic images (not contained in the training set). (See Fig. 5.) The average accuracy for classifying whether a 2-d image patch is a projection of a grasping point was 94.2% (evaluated on a balanced test set), although the accuracy in predicting 3-d grasping points was higher because the probabilistic model for inferring a 3-d grasping point automatically aggregates data from multiple images, and therefore "fixes" some of the errors from individual classifiers.

We then tested the algorithm on the physical robotic arm. Here, the task was to use input from a web-camera, mounted on the robot, to pick up an object placed in front of the robot. Recall that the parameters of the vision algorithm were trained from synthetic images of a small set of six object classes, namely books, martini glasses, white-board erasers, coffee mugs, tea cups and pencils. We performed experiments on coffee mugs, wine glasses (partially filled with water), pencils, books, and erasers—but all of different dimensions and appearance than the ones in the training set— as well as a large set of objects from novel object classes, such as rolls of duct tape, markers, a translucent box, jugs, knife-cutters, cellphones, pens, keys, screwdrivers, staplers, toothbrushes, a thick coil of wire, a strangely shaped power horn, etc. (See Fig. 1.) We note that many of these objects are translucent, textureless, and/or reflective, making 3-d reconstruction difficult for standard stereo systems. (Indeed, a carefully-calibrated Point Gray stereo system, the Bumblebee BB-COL-20, —with higher quality cameras than our web-camera—fails to accurately reconstruct the visible portions of 9 out of 12 objects.)

In extensive experiments, the algorithm for predicting grasps in images appeared to generalize very well. Despite being tested on images of real (rather than synthetic) objects, including many very different from ones in the training set, it was usually able to identify correct grasp points. We note that test set error (in terms of average absolute error in the predicted position of the grasp point) on the real images was only somewhat higher than the error on synthetic images; this shows that the algorithm trained on synthetic images transfers well to real images. (Over all 5 object types used in the synthetic data, average absolute error was 0.81cm in the synthetic images; and over all the 13 real test objects, average error was 1.83cm.) For comparison, neonate humans can grasp simple objects with an average accuracy of 1.5cm. [2]

Table 1 shows the errors in the predicted grasping points on the test set. The table presents results separately for objects which were similar to those we trained on (e.g., coffee mugs) and those which were very dissimilar to the training objects (e.g., duct tape). In addition to reporting errors in grasp positions, we also report the grasp success rate, i.e., the fraction of times the robotic arm was able to physically pick up the object (out of 4 trials). On average, the robot picked up the novel objects 87.5% of the time.

For simple objects such as cellphones, wine glasses, keys, toothbrushes, etc., the algorithm performed perfectly (100% grasp success rate). However, grasping objects such as mugs or jugs (by the handle) allows only a narrow trajectory of approach—where one "finger" is inserted into the handle—so that even a small error in the grasping point identification causes the arm to hit and move the object, resulting in a failed grasp attempt. Although it may be possible to improve the

algorithm's accuracy, we believe that these problems can best be solved by using a more advanced robotic arm that is capable of haptic (touch) feedback.

In many instances, the algorithm was able to pick up completely novel objects (a strangely shaped power-horn, duct-tape, solder tool holder, etc.; see Fig. 1 and 6). Perceiving a transparent wine glass is a difficult problem for standard vision (e.g., stereopsis) algorithms because of reflections, etc. However, as shown in Table 1, our algorithm successfully picked it up 100% of the time. The same rate of success holds even if the glass is 2/3 filled with water. Videos showing the robot grasping the objects, are available at

**http://ai.stanford.edu/∼asaxena/learninggrasp/**

We also applied our learning algorithm to the task of unloading items from a dishwasher.[9] Fig. 5 demonstrates that the algorithm correctly identifies the grasp on multiple objects even in the presence of clutter and occlusion. Fig. 6 shows our robot unloading some items from a dishwasher.

## 5   Conclusions

We proposed an algorithm to enable a robot to grasp an object that it has never seen before. Our learning algorithm neither tries to build, nor requires, a 3-d model of the object. Instead it predicts, directly as a function of the images, a point at which to grasp the object. In our experiments, the algorithm generalizes very well to novel objects and environments, and our robot successfully grasped a wide variety of objects, such as wine glasses, duct tape, markers, a translucent box, jugs, knife-cutters, cellphones, keys, screwdrivers, staplers, toothbrushes, a thick coil of wire, a strangely shaped power horn, and others, none of which were seen in the training set.

**Acknowledgment**

We give warm thanks to Anya Petrovskaya, Morgan Quigley, and Jimmy Zhang for help with the robotic arm control driver software. This work was supported by the DARPA transfer learning program under contract number FA8750-05-2-0249.

## Footnotes

[1]For example, picking up a heavy book lying flat on table might require a sequence of complex manipulations, such as to first slide the book slightly past the edge of the table so that the manipulator can place its fingers around the book.

[2]An earlier version of this work without the probabilistic model and using simpler learning/inference was described in [12].

[3]We use YCbCr color space, where Y is the intensity channel, and Cb and Cr are color channels.

[4]Ray tracing [3] is a standard image rendering method from computer graphics. It handles many real-world optical phenomenon such as multiple specular reflections, textures, soft shadows, smooth curves, and caustics. We used PovRay, an open source ray tracer.

[5]There is a relation between the quality of the synthetically generated images and the accuracy of the algorithm. The better the quality of the synthetically generated images and graphical realism, the better the accuracy of the algorithm. Therefore, we use a ray tracer instead of faster, but cruder, openGL style graphics. Michels, Saxena and Ng [6] used synthetic openGL images to learn distances in natural scenes. However, because openGL style graphics have less realism, the learning performance sometimes *decreased* with added complexity in the scenes.

[6]The robot position/orientation error is typically small (position is usually accurate to within 1mm), but it is still important to model this error. From our experiments (see Section 4), if we set $\sigma^2 = 0$, the triangulation is highly inaccurate, with average error in predicting grasping point being $15.40$ cm, as compared to $1.85$ cm when appropriate $\sigma^2$ is chosen.

[7]Since there are only a few places in an image where $P(z_i(u,v) = 1|C_i) > 0$, the counting algorithm is computationally much less expensive than enumerating over all grid-cells. In practice, we found that restricting attention to areas where $P(z_i(u,v) = 1|C_i) > 0.1$ allows us to further reduce the number of rays to be considered, with no noticeable degradation in performance.

[8]The approach position is set to be a fixed distance away from the predicted grasp point.

[9]To improve performance, we also used depth-based features. More formally, we applied our texture based features to the depth image obtained from a stereo camera, and appended them to the feature vector used in classification. We also appended some hand-labeled real examples of dishwasher images to the training set to prevent the algorithm from identifying grasping points on background clutter, such as dishwasher prongs.

## References

[1] A. Bicchi and V. Kumar. Robotic grasping and contact: a review. In *ICRA*, 2000.

[2] T. G. R. Bower, J. M. Broughton, and M. K. Moore. Demonstration of intention in the reaching behaviour of neonate humans. *Nature*, 228:679–681, 1970.

[3] A. S. Glassner. *An Introduction to Ray Tracing*. Morgan Kaufmann Publishers, Inc., San Francisco, 1989.

[4] K. Hsiao and T. Lozano-Perez. Imitation learning of whole-body grasps. In *IEEE/RJS International Conference on Intelligent Robots and Systems (IROS)*, 2006.

[5] M. T. Mason and J. K. Salisbury. Manipulator grasping and pushing operations. In *Robot Hands and the Mechanics of Manipulation*. The MIT Press, Cambridge, MA, 1985.

[6] J. Michels, A. Saxena, and A. Y. Ng. High speed obstacle avoidance using monocular vision and reinforcement learning. In *ICML*, 2005.

[7] Miller and et. al. Automatic grasp planning using shape primitives. In *ICRA*, 2003.

[8] R. Pelossof and et. al. An svm learning approach to robotic grasping. In *ICRA*, 2004.

[9] J. H. Piater. Learning visual features to predict hand orientations. In *ICML Workshop on Machine Learning of Spatial Knowledge*, 2002.

[10] R. Platt, A. H. Fagg, and R. Grupen. Reusing schematic grasping policies. In *IEEE-RAS International Conference on Humanoid Robots, Tsukuba, Japan*, 2005.

[11] A. Saxena, S. H. Chung, and A. Y. Ng. Learning depth from single monocular images. In *NIPS 18*, 2005.

[12] A. Saxena, J. Driemeyer, J. Kearns, C. Osondu, and A. Y. Ng. Learning to grasp novel objects using vision. In *10th International Symposium of Experimental Robotics (ISER)*, 2006.

[13] A. Saxena, J. Schulte, and A. Y. Ng. Depth estimation using monocular and stereo cues. In *20th International Joint Conference on Artificial Intelligence (IJCAI)*, 2007.

[14] H. Schneiderman and T. Kanade. Probabilistic modeling of local appearance and spatial relationships for object recognition. In *CVPR*, 1998.

[15] T. Shin-ichi and M. Satoshi. Living and working with robots. *Nipponia*, 2000.

